# Online Learning for Latent Dirichlet Allocation

**Matthew D. Hoffman**
Department of Computer Science
Princeton University
Princeton, NJ
mdhoffma@cs.princeton.edu

**David M. Blei**
Department of Computer Science
Princeton University
Princeton, NJ
blei@cs.princeton.edu

**Francis Bach**
INRIA—Ecole Normale Supérieure
Paris, France
francis.bach@ens.fr

## Abstract

We develop an online variational Bayes (VB) algorithm for Latent Dirichlet Allocation (LDA). Online LDA is based on online stochastic optimization with a natural gradient step, which we show converges to a local optimum of the VB objective function. It can handily analyze massive document collections, including those arriving in a stream. We study the performance of online LDA in several ways, including by fitting a 100-topic topic model to 3.3M articles from Wikipedia in a single pass. We demonstrate that online LDA finds topic models as good or better than those found with batch VB, and in a fraction of the time.

## 1 Introduction

Hierarchical Bayesian modeling has become a mainstay in machine learning and applied statistics. Bayesian models provide a natural way to encode assumptions about observed data, and analysis proceeds by examining the posterior distribution of model parameters and latent variables conditioned on a set of observations. For example, research in probabilistic topic modeling—the application we will focus on in this paper—revolves around fitting complex hierarchical Bayesian models to large collections of documents. In a topic model, the posterior distribution reveals latent semantic structure that can be used for many applications.

For topic models and many other Bayesian models of interest, however, the posterior is intractable to compute and researchers must appeal to approximate posterior inference. Modern approximate posterior inference algorithms fall in two categories—sampling approaches and optimization approaches. Sampling approaches are usually based on Markov Chain Monte Carlo (MCMC) sampling, where a Markov chain is defined whose stationary distribution is the posterior of interest. Optimization approaches are usually based on variational inference, which is called variational Bayes (VB) when used in a Bayesian hierarchical model. Whereas MCMC methods seek to generate independent samples from the posterior, VB optimizes a simplified parametric distribution to be close in Kullback-Leibler divergence to the posterior. Although the choice of approximate posterior introduces bias, VB is empirically shown to be faster than and as accurate as MCMC, which makes it an attractive option when applying Bayesian models to large datasets [1, 2, 3].

Nonetheless, large scale data analysis with VB can be computationally difficult. Standard "batch" VB algorithms iterate between analyzing each observation and updating dataset-wide variational parameters. The per-iteration cost of batch algorithms can quickly become impractical for very large datasets. In topic modeling applications, this issue is particularly relevant—topic modeling promises

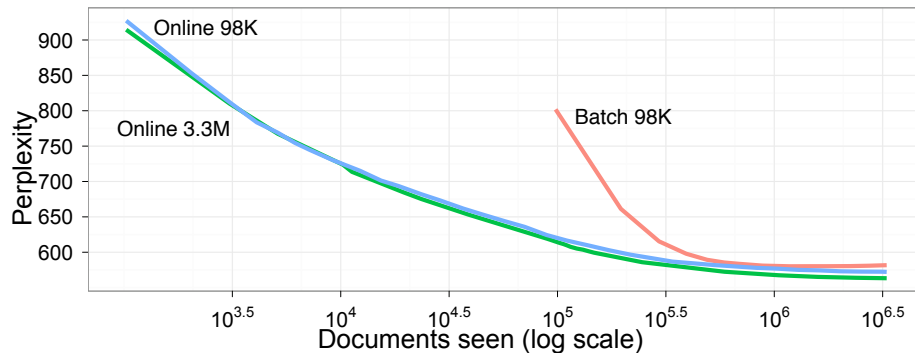

| Documents analyzed | 2048 | 4096 | 8192 | 12288 | 16384 | 32768 | 49152 | 65536 |
|---|---|---|---|---|---|---|---|---|
| **Top eight words** | systems | systems | service | service | service | business | business | business |
| | road | health | systems | systems | companies | service | service | industry |
| | made | communication | health | companies | systems | companies | companies | service |
| | service | service | companies | business | business | industry | industry | companies |
| | announced | billion | market | company | company | company | services | services |
| | national | language | communication | billion | industry | management | company | company |
| | west | care | company | health | market | systems | management | management |
| | language | road | billion | industry | billion | services | public | public |

Figure 1: Top: Perplexity on held-out Wikipedia documents as a function of number of documents analyzed, i.e., the number of E steps. Online VB run on 3.3 million unique Wikipedia articles is compared with online VB run on 98,000 Wikipedia articles and with the batch algorithm run on the same 98,000 articles. The online algorithms converge much faster than the batch algorithm does. Bottom: Evolution of a topic about business as online LDA sees more and more documents.

to summarize the latent structure of massive document collections that cannot be annotated by hand. A central research problem for topic modeling is to efficiently fit models to larger corpora [4, 5].

To this end, we develop an online variational Bayes algorithm for latent Dirichlet allocation (LDA), one of the simplest topic models and one on which many others are based. Our algorithm is based on online stochastic optimization, which has been shown to produce good parameter estimates dramatically faster than batch algorithms on large datasets [6]. Online LDA handily analyzes massive collections of documents and, moreover, online LDA need not locally store or collect the documents— each can arrive in a stream and be discarded after one look.

In the subsequent sections, we derive online LDA and show that it converges to a stationary point of the variational objective function. We study the performance of online LDA in several ways, including by fitting a topic model to 3.3M articles from Wikipedia without looking at the same article twice. We show that online LDA finds topic models as good as or better than those found with batch VB, and in a fraction of the time (see figure 1). Online variational Bayes is a practical new method for estimating the posterior of complex hierarchical Bayesian models.

## 2   Online variational Bayes for latent Dirichlet allocation

Latent Dirichlet Allocation (LDA) [7] is a Bayesian probabilistic model of text documents. It assumes a collection of $K$ "topics." Each topic defines a multinomial distribution over the vocabulary and is assumed to have been drawn from a Dirichlet, $\beta_k \sim \text{Dirichlet}(\eta)$. Given the topics, LDA assumes the following generative process for each document $d$. First, draw a distribution over topics $\theta_d \sim \text{Dirichlet}(\alpha)$. Then, for each word $i$ in the document, draw a topic index $z_{di} \in \{1, \ldots, K\}$ from the topic weights $z_{di} \sim \theta_d$ and draw the observed word $w_{di}$ from the selected topic, $w_{di} \sim \beta_{z_{di}}$. For simplicity, we assume symmetric priors on $\theta$ and $\beta$, but this assumption is easy to relax [8].

Note that if we sum over the topic assignments $z$, then we get $p(w_{di}|\theta_d, \beta) = \sum_k \theta_{dk} \beta_{kw}$. This leads to the "multinomial PCA" interpretation of LDA; we can think of LDA as a probabilistic factorization of the matrix of word counts $n$ (where $n_{dw}$ is the number of times word $w$ appears in document $d$) into a matrix of topic weights $\theta$ and a dictionary of topics $\beta$ [9]. Our work can thus

be seen as an extension of online matrix factorization techniques that optimize squared error [10] to more general probabilistic formulations.

We can analyze a corpus of documents with LDA by examining the posterior distribution of the topics $\boldsymbol{\beta}$, topic proportions $\boldsymbol{\theta}$, and topic assignments $\boldsymbol{z}$ conditioned on the documents. This reveals latent structure in the collection that can be used for prediction or data exploration. This posterior cannot be computed directly [7], and is usually approximated using Markov Chain Monte Carlo (MCMC) methods or variational inference. Both classes of methods are effective, but both present significant computational challenges in the face of massive data sets.Developing scalable approximate inference methods for topic models is an active area of research [3, 4, 5, 11].

To this end, we develop online variational inference for LDA, an approximate posterior inference algorithm that can analyze massive collections of documents. We first review the traditional variational Bayes algorithm for LDA and its objective function, then present our online method, and show that it converges to a stationary point of the same objective function.

## 2.1 Batch variational Bayes for LDA

In Variational Bayesian inference (VB) the true posterior is approximated by a simpler distribution $q(\boldsymbol{z}, \boldsymbol{\theta}, \boldsymbol{\beta})$, which is indexed by a set of free parameters [12, 13]. These parameters are optimized to maximize the Evidence Lower BOund (ELBO):

$$\log p(\boldsymbol{w}|\alpha, \eta) \geq \mathcal{L}(\boldsymbol{w}, \boldsymbol{\phi}, \boldsymbol{\gamma}, \boldsymbol{\lambda}) \triangleq \mathbb{E}_q[\log p(\boldsymbol{w}, \boldsymbol{z}, \boldsymbol{\theta}, \boldsymbol{\beta}|\alpha, \eta)] - \mathbb{E}_q[\log q(\boldsymbol{z}, \boldsymbol{\theta}, \boldsymbol{\beta})]. \quad (1)$$

Maximizing the ELBO is equivalent to minimizing the KL divergence between $q(\boldsymbol{z}, \boldsymbol{\theta}, \boldsymbol{\beta})$ and the posterior $p(\boldsymbol{z}, \boldsymbol{\theta}, \boldsymbol{\beta}|\boldsymbol{w}, \alpha, \eta)$. Following [7], we choose a fully factorized distribution $q$ of the form

$$q(z_{di} = k) = \phi_{dw_{di}k}; \quad q(\theta_d) = \text{Dirichlet}(\theta_d; \gamma_d); \quad q(\beta_k) = \text{Dirichlet}(\beta_k; \lambda_k), \quad (2)$$

The posterior over the per-word topic assignments $z$ is parameterized by $\phi$, the posterior over the per-document topic weights $\theta$ is parameterized by $\theta$, and the posterior over the topics $\beta$ is parameterized by $\lambda$. As a shorthand, we refer to $\lambda$ as "the topics." Equation 1 factorizes to

$$\begin{aligned}\mathcal{L}(\boldsymbol{w}, \boldsymbol{\phi}, \boldsymbol{\gamma}, \boldsymbol{\lambda}) = \sum_d \big\{ &\mathbb{E}_q[\log p(w_d|\theta_d, z_d, \boldsymbol{\beta})] + \mathbb{E}_q[\log p(z_d|\theta_d)] - \mathbb{E}_q[\log q(z_d)] \\ &+ \mathbb{E}_q[\log p(\theta_d|\alpha)] - \mathbb{E}_q[\log q(\theta_d)] + (\mathbb{E}_q[\log p(\boldsymbol{\beta}|\eta)] - \mathbb{E}_q[\log q(\boldsymbol{\beta})])/D \big\}.\end{aligned} \quad (3)$$

Notice we have brought the per-corpus terms into the summation over documents, and divided them by the number of documents $D$. This step will help us to derive an online inference algorithm.

We now expand the expectations above to be functions of the variational parameters. This reveals that the variational objective relies only on $n_{dw}$, the number of times word $w$ appears in document $d$. When using VB—as opposed to MCMC—documents can be summarized by their word counts,

$$\begin{aligned}\mathcal{L} = \sum_d \sum_w n_{dw} \sum_k & \phi_{dwk}(\mathbb{E}_q[\log \theta_{dk}] + \mathbb{E}_q[\log \beta_{kw}] - \log \phi_{dwk}) \\ & - \log \Gamma(\sum_k \gamma_{dk}) + \sum_k (\alpha - \gamma_{dk})\mathbb{E}_q[\log \theta_{dk}] + \log \Gamma(\gamma_{dk}) \\ & + (\sum_k -\log \Gamma(\sum_w \lambda_{kw}) + \sum_w(\eta - \lambda_{kw})\mathbb{E}_q[\log \beta_{kw}] + \log \Gamma(\lambda_{kw}))/D \\ & + \log \Gamma(K\alpha) - K \log \Gamma(\alpha) + (\log \Gamma(W\eta) - W \log \Gamma(\eta))/D \\ \triangleq \sum_d & \ell(n_d, \phi_d, \gamma_d, \boldsymbol{\lambda}),\end{aligned} \quad (4)$$

where $W$ is the size of the vocabulary and $D$ is the number of documents. $\ell(n_d, \phi_d, \gamma_d, \boldsymbol{\lambda})$ denotes the contribution of document $d$ to the ELBO.

$\mathcal{L}$ can be optimized using coordinate ascent over the variational parameters $\boldsymbol{\phi}, \boldsymbol{\gamma}, \boldsymbol{\lambda}$ [7]:

$$\phi_{dwk} \propto \exp\{\mathbb{E}_q[\log \theta_{dk}] + \mathbb{E}_q[\log \beta_{kw}]\}; \quad \gamma_{dk} = \alpha + \sum_w n_{dw}\phi_{dwk}; \quad \lambda_{kw} = \eta + \sum_d n_{dw}\phi_{dwk}. \quad (5)$$

The expectations under $q$ of $\log \boldsymbol{\theta}$ and $\log \boldsymbol{\beta}$ are

$$\mathbb{E}_q[\log \theta_{dk}] = \Psi(\gamma_{dk}) - \Psi(\sum_{i=1}^K \gamma_{di}); \quad \mathbb{E}_q[\log \beta_{kw}] = \Psi(\lambda_{kw}) - \Psi(\sum_{i=1}^W \lambda_{ki}), \quad (6)$$

where $\Psi$ denotes the digamma function (the first derivative of the logarithm of the gamma function).

The updates in equation 5 are guaranteed to converge to a stationary point of the ELBO. By analogy to the Expectation-Maximization (EM) algorithm [14], we can partition these updates into an "E" step—iteratively updating $\boldsymbol{\gamma}$ and $\boldsymbol{\phi}$ until convergence, holding $\boldsymbol{\lambda}$ fixed—and an "M" step—updating $\boldsymbol{\lambda}$ given $\boldsymbol{\phi}$. In practice, this algorithm converges to a better solution if we reinitialize $\boldsymbol{\gamma}$ and $\boldsymbol{\phi}$ before each E step. Algorithm 1 outlines batch VB for LDA.

---

**Algorithm 1** Batch variational Bayes for LDA

> Initialize $\boldsymbol{\lambda}$ randomly.
> **while** relative improvement in $\mathcal{L}(\boldsymbol{w}, \boldsymbol{\phi}, \boldsymbol{\gamma}, \boldsymbol{\lambda}) > 0.00001$ **do**
>   *E step*:
>   **for** $d = 1$ to $D$ **do**
>     Initialize $\gamma_{dk} = 1$. (The constant 1 is arbitrary.)
>     **repeat**
>       Set $\phi_{dwk} \propto \exp\{\mathbb{E}_q[\log \theta_{dk}] + \mathbb{E}_q[\log \beta_{kw}]\}$
>       Set $\gamma_{dk} = \alpha + \sum_w \phi_{dwk} n_{dw}$
>     **until** $\frac{1}{K} \sum_k |\text{change in} \gamma_{dk}| < 0.00001$
>   **end for**
>   *M step*:
>   Set $\lambda_{kw} = \eta + \sum_d n_{dw} \phi_{dwk}$
> **end while**

---

## 2.2 Online variational inference for LDA

Algorithm 1 has constant memory requirements and empirically converges faster than batch collapsed Gibbs sampling [3]. However, it still requires a full pass through the entire corpus each iteration. It can therefore be slow to apply to very large datasets, and is not naturally suited to settings where new data is constantly arriving. We propose an online variational inference algorithm for fitting $\boldsymbol{\lambda}$, the parameters to the variational posterior over the topic distributions $\boldsymbol{\beta}$. Our algorithm is nearly as simple as the batch VB algorithm, but converges much faster for large datasets.

A good setting of the topics $\boldsymbol{\lambda}$ is one for which the ELBO $\mathcal{L}$ is as high as possible after fitting the per-document variational parameters $\boldsymbol{\gamma}$ and $\boldsymbol{\phi}$ with the E step defined in algorithm 1. Let $\gamma(n_d, \boldsymbol{\lambda})$ and $\phi(n_d, \boldsymbol{\lambda})$ be the values of $\gamma_d$ and $\phi_d$ produced by the E step. Our goal is to set $\boldsymbol{\lambda}$ to maximize

$$\mathcal{L}(\boldsymbol{n}, \boldsymbol{\lambda}) \triangleq \sum_d \ell(n_d, \gamma(n_d, \boldsymbol{\lambda}), \phi(n_d, \boldsymbol{\lambda}), \boldsymbol{\lambda}), \tag{7}$$

where $\ell(n_d, \gamma_d, \phi_d, \boldsymbol{\lambda})$ is the $d$th document's contribution to the variational bound in equation 4. This is analogous to the goal of least-squares matrix factorization, although the ELBO for LDA is less convenient to work with than a simple squared loss function such as the one in [10].

Online VB for LDA ("online LDA") is described in algorithm 2. As the $t$th vector of word counts $n_t$ is observed, we perform an E step to find locally optimal values of $\gamma_t$ and $\phi_t$, holding $\boldsymbol{\lambda}$ fixed. We then compute $\tilde{\boldsymbol{\lambda}}$, the setting of $\boldsymbol{\lambda}$ that would be optimal (given $\phi_t$) if our entire corpus consisted of the single document $n_t$ repeated $D$ times. $D$ is the number of unique documents available to the algorithm, e.g. the size of a corpus. (In the true online case $D \to \infty$, corresponding to empirical Bayes estimation of $\boldsymbol{\beta}$.) We then update $\boldsymbol{\lambda}$ using a weighted average of its previous value and $\tilde{\boldsymbol{\lambda}}$. The weight given to $\tilde{\boldsymbol{\lambda}}$ is given by $\rho_t \triangleq (\tau_0 + t)^{-\kappa}$, where $\kappa \in (0.5, 1]$ controls the rate at which old values of $\tilde{\boldsymbol{\lambda}}$ are forgotten and $\tau_0 \geq 0$ slows down the early iterations of the algorithm. The condition that $\kappa \in (0.5, 1]$ is needed to guarantee convergence. We show in section 2.3 that online LDA corresponds to a stochastic natural gradient algorithm on the variational objective $\mathcal{L}$ [15, 16].

This algorithm closely resembles one proposed in [16] for online VB on models with hidden data—the most important difference is that we use an approximate E step to optimize $\gamma_t$ and $\phi_t$, since we cannot compute the conditional distribution $p(z_t, \theta_t | \boldsymbol{\beta}, n_t, \alpha)$ exactly.

**Mini-batches.** A common technique in stochastic learning is to consider multiple observations per update to reduce noise [6, 17]. In online LDA, this means computing $\tilde{\boldsymbol{\lambda}}$ using $S > 1$ observations:

$$\tilde{\boldsymbol{\lambda}}_{kw} = \eta + \frac{D}{S} \sum_s n_{tsk} \phi_{tskw}, \tag{8}$$

where $n_{ts}$ is the $s$th document in mini-batch $t$. The variational parameters $\phi_{ts}$ and $\gamma_{ts}$ for this document are fit with a normal E step. Note that we recover batch VB when $S = D$ and $\kappa = 0$.

**Hyperparameter estimation.** In batch variational LDA, point estimates of the hyperparameters $\alpha$ and $\eta$ can be fit given $\boldsymbol{\gamma}$ and $\boldsymbol{\lambda}$ using a linear-time Newton-Raphson method [7]. We can likewise

**Algorithm 2** Online variational Bayes for LDA

---
Define $\rho_t \triangleq (\tau_0 + t)^{-\kappa}$
Initialize $\boldsymbol{\lambda}$ randomly.
**for** $t = 0$ to $\infty$ **do**
   *E step*:
   Initialize $\gamma_{tk} = 1$. (The constant 1 is arbitrary.)
   **repeat**
      Set $\phi_{twk} \propto \exp\{\mathbb{E}_q[\log \theta_{tk}] + \mathbb{E}_q[\log \beta_{kw}]\}$
      Set $\gamma_{tk} = \alpha + \sum_w \phi_{twk} n_{tw}$
   **until** $\frac{1}{K}\sum_k |\text{change in}\gamma_{tk}| < 0.00001$
   *M step*:
   Compute $\tilde{\lambda}_{kw} = \eta + D n_{tw}\phi_{twk}$
   Set $\boldsymbol{\lambda} = (1 - \rho_t)\boldsymbol{\lambda} + \rho_t \tilde{\boldsymbol{\lambda}}$.
**end for**

---

incorporate updates for $\alpha$ and $\eta$ into online LDA:

$$\alpha \leftarrow \alpha - \rho_t \tilde{\alpha}(\gamma_t); \quad \eta \leftarrow \eta - \rho_t \tilde{\eta}(\boldsymbol{\lambda}), \tag{9}$$

where $\tilde{\alpha}(\gamma_t)$ is the inverse of the Hessian times the gradient $\nabla_\alpha \ell(n_t, \gamma_t, \phi_t, \boldsymbol{\lambda})$, $\tilde{\eta}(\boldsymbol{\lambda})$ is the inverse of the Hessian times the gradient $\nabla_\eta \mathcal{L}$, and $\rho_t \triangleq (\tau_0 + t)^{-\kappa}$ as elsewhere.

## 2.3 Analysis of convergence

In this section we show that algorithm 2 converges to a stationary point of the objective defined in equation 7. Since variational inference replaces sampling with optimization, we can use results from stochastic optimization to analyze online LDA. Stochastic optimization algorithms optimize an objective using noisy estimates of its gradient [18]. Although there is no explicit gradient computation, algorithm 2 can be interpreted as a stochastic natural gradient algorithm [16, 15].

We begin by deriving a related first-order stochastic gradient algorithm for LDA. Let $g(n)$ denote the population distribution over documents $n$ from which we will repeatedly sample documents:

$$g(n) \triangleq \frac{1}{D}\sum_{d=1}^{D} \mathbb{I}[n = n_d]. \tag{10}$$

$\mathbb{I}[n = n_d]$ is 1 if $n = n_d$ and 0 otherwise. If this population consists of the $D$ documents in the corpus, then we can rewrite equation 7 as

$$\mathcal{L}(g, \boldsymbol{\lambda}) \triangleq D\mathbb{E}_g[\ell(n, \gamma(n, \boldsymbol{\lambda}), \phi(n, \boldsymbol{\lambda}), \boldsymbol{\lambda})|\boldsymbol{\lambda}]. \tag{11}$$

where $\ell$ is defined as in equation 3. We can optimize equation 11 over $\boldsymbol{\lambda}$ by repeatedly drawing an observation $n_t \sim g$, computing $\gamma_t \triangleq \gamma(n_t, \boldsymbol{\lambda})$ and $\phi_t \triangleq \phi(n_t, \boldsymbol{\lambda})$, and applying the update

$$\boldsymbol{\lambda} \leftarrow \boldsymbol{\lambda} + \rho_t D \nabla_\lambda \ell(n_t, \gamma_t, \phi_t, \boldsymbol{\lambda}) \tag{12}$$

where $\rho_t \triangleq (\tau_0 + t)^{-\kappa}$ as in algorithm 2. If we condition on the current value of $\boldsymbol{\lambda}$ and treat $\gamma_t$ and $\phi_t$ as random variables drawn at the same time as each observed document $n_t$, then $\mathbb{E}_g[D\nabla_\lambda \ell(n_t, \gamma_t, \phi_t, \boldsymbol{\lambda})|\boldsymbol{\lambda}] = \nabla_\lambda \sum_d \ell(n_d, \gamma_d, \phi_d, \boldsymbol{\lambda})$. Thus, since $\sum_{t=0}^{\infty} \rho_t = \infty$ and $\sum_{t=0}^{\infty} \rho_t^2 < \infty$, the analysis in [19] shows both that $\boldsymbol{\lambda}$ converges and that the gradient $\nabla_\lambda \sum_d \ell(n_d, \gamma_d, \phi_d, \boldsymbol{\lambda})$ converges to 0, and thus that $\boldsymbol{\lambda}$ converges to a stationary point.[1]

The update in equation 12 only makes use of first-order gradient information. Stochastic gradient algorithms can be sped up by multiplying the gradient by the inverse of an appropriate positive definite matrix $H$ [19]. One choice for $H$ is the Hessian of the objective function. In variational inference, an alternative is to use the Fisher information matrix of the variational distribution $q$ (i.e., the Hessian of the log of the variational probability density function), which corresponds to using

a natural gradient method instead of a (quasi-) Newton method [16, 15]. Following the analysis in [16], the gradient of the per-document ELBO $\ell$ can be written as

$$
\begin{aligned}
\frac{\partial \ell(n_t, \gamma_t, \phi_t, \lambda)}{\partial \lambda_{kw}} &= \sum_{v=1}^{W} \frac{\partial \mathbb{E}_q[\log \beta_{kv}]}{\partial \lambda_{kw}} (-\lambda_{kv}/D + \eta/D + n_{tv}\phi_{tvk}) \\
&= \sum_{v=1}^{W} -\frac{\partial^2 \log q(\beta_k)}{\partial \lambda_{kv} \partial \lambda_{kw}} (-\lambda_{kv}/D + \eta/D + n_{tv}\phi_{tvk}),
\end{aligned}
\tag{13}
$$

where we have used the fact that $\mathbb{E}_q[\log \beta_{kv}]$ is the derivative of the log-normalizer of $q(\log \beta_k)$. By definition, multiplying equation 13 by the inverse of the Fisher information matrix yields

$$
\left[ \left( -\frac{\partial^2 \log q(\log \beta_k)}{\partial \lambda_k \partial \lambda_k^T} \right)^{-1} \frac{\partial \ell(n_t, \gamma_t, \phi_t, \lambda)}{\partial \lambda_k} \right]_w = -\lambda_{kw}/D + \eta/D + n_{tw}\phi_{twk}.
\tag{14}
$$

Multiplying equation 14 by $\rho_t D$ and adding it to $\lambda_{kw}$ yields the update for $\boldsymbol{\lambda}$ in algorithm 2. Thus we can interpret our algorithm as a stochastic natural gradient algorithm, as in [16].

## 3   Related work

**Comparison with other stochastic learning algorithms.**   In the standard stochastic gradient optimization setup, the number of parameters to be fit does not depend on the number of observations [19]. However, some learning algorithms must also fit a set of per-observation parameters (such as the per-document variational parameters $\gamma_d$ and $\phi_d$ in LDA). The problem is addressed by online coordinate ascent algorithms such as those described in [20, 21, 16, 17, 10]. The goal of these algorithms is to set the global parameters so that the objective is as good as possible once the per-observation parameters are optimized. Most of these approaches assume the computability of a unique optimum for the per-observation parameters, which is not available for LDA.

**Efficient sampling methods.**   Markov Chain Monte Carlo (MCMC) methods form one class of approximate inference algorithms for LDA. Collapsed Gibbs Sampling (CGS) is a popular MCMC approach that samples from the posterior over topic assignments $\boldsymbol{z}$ by repeatedly sampling the topic assignment $z_{di}$ conditioned on the data and all other topic assignments [22].

One online MCMC approach adapts CGS by sampling topic assignments $z_{di}$ based on the topic assignments and data for all *previously analyzed* words, instead of all other words in the corpus [23]. This algorithm is fast and has constant memory requirements, but is not guaranteed to converge to the posterior. Two alternative online MCMC approaches were considered in [24]. The first, called incremental LDA, periodically resamples the topic assignments for previously analyzed words. The second approach uses particle filtering instead of CGS. In a study in [24], none of these three online MCMC algorithms performed as well as batch CGS.

Instead of online methods, the authors of [4] used parallel computing to apply LDA to large corpora. They developed two approximate parallel CGS schemes for LDA that gave similar predictive performance on held-out documents to batch CGS. However, they require parallel hardware, and their complexity and memory costs still scale linearly with the number of documents.

Except for the algorithm in [23] (which is not guaranteed to converge), all of the MCMC algorithms described above have memory costs that scale linearly with the number of documents analyzed. By contrast, batch VB can be implemented using constant memory, and parallelizes easily. As we will show in the next section, its online counterpart is even faster.

## 4   Experiments

We ran several experiments to evaluate online LDA's efficiency and effectiveness. The first set of experiments compares algorithms 1 and 2 on static datasets. The second set of experiments evaluates online VB in the setting where new documents are constantly being observed. Both algorithms were implemented in Python using Numpy. The implementations are as similar as possible.[2]

Table 1: Best settings of $\kappa$ and $\tau_0$ for various mini-batch sizes $S$, with resulting perplexities on *Nature* and Wikipedia corpora.

| Best parameter settings for *Nature* corpus | | | | | | | |
|---|---|---|---|---|---|---|---|
| $S$ | 1 | 4 | 16 | 64 | 256 | 1024 | 4096 | 16384 |
| $\kappa$ | 0.9 | 0.8 | 0.8 | 0.7 | 0.6 | 0.5 | 0.5 | 0.5 |
| $\tau_0$ | 1024 | 1024 | 1024 | 1024 | 1024 | 256 | 64 | 1 |
| Perplexity | 1132 | 1087 | 1052 | 1053 | 1042 | 1031 | 1030 | 1046 |

| Best parameter settings for Wikipedia corpus | | | | | | | |
|---|---|---|---|---|---|---|---|
| $S$ | 1 | 4 | 16 | 64 | 256 | 1024 | 4096 | 16384 |
| $\kappa$ | 0.9 | 0.9 | 0.8 | 0.7 | 0.6 | 0.5 | 0.5 | 0.5 |
| $\tau_0$ | 1024 | 1024 | 1024 | 1024 | 1024 | 1024 | 64 | 1 |
| Perplexity | 675 | 640 | 611 | 595 | 588 | 584 | 580 | 584 |

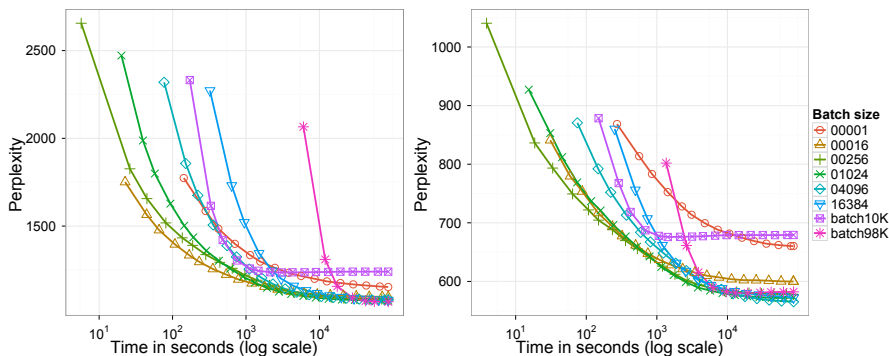

Figure 2: Held-out perplexity obtained on the *Nature* (left) and Wikipedia (right) corpora as a function of CPU time. For moderately large mini-batch sizes, online LDA finds solutions as good as those that the batch LDA finds, but with much less computation. When fit to a 10,000-document subset of the training corpus batch LDA's speed improves, but its performance suffers.

We use perplexity on held-out data as a measure of model fit. Perplexity is defined as the geometric mean of the inverse marginal probability of each word in the held-out set of documents:

$$\text{perplexity}(\boldsymbol{n}^{\text{test}}, \boldsymbol{\lambda}, \alpha) \triangleq \exp\left\{-\left(\textstyle\sum_i \log p(n_i^{\text{test}}|\alpha, \boldsymbol{\beta})\right)/\left(\textstyle\sum_{i,w} n_{iw}^{\text{test}}\right)\right\} \qquad (15)$$

where $n_i^{\text{test}}$ denotes the vector of word counts for the $i$th document. Since we cannot directly compute $\log p(n_i^{\text{test}}|\alpha, \boldsymbol{\beta})$, we use a lower bound on perplexity as a proxy:

$$\text{perplexity}(\boldsymbol{n}^{\text{test}}, \boldsymbol{\lambda}, \alpha) \leq \exp\left\{-\left(\textstyle\sum_i \mathbb{E}_q[\log p(n_i^{\text{test}}, \theta_i, z_i|\alpha, \boldsymbol{\beta})] - \mathbb{E}_q[\log q(\theta_i, z_i)]\right)\left(\textstyle\sum_{i,w} n_{iw}^{\text{test}}\right)\right\}. \qquad (16)$$

The per-document parameters $\gamma_i$ and $\phi_i$ for the variational distributions $q(\theta_i)$ and $q(z_i)$ are fit using the E step in algorithm 2. The topics $\boldsymbol{\lambda}$ are fit to a training set of documents and then held fixed. In all experiments $\alpha$ and $\eta$ are fixed at 0.01 and the number of topics $K = 100$.

There is some question as to the meaningfulness of perplexity as a metric for comparing different topic models [25]. Held-out likelihood metrics are nonetheless well suited to measuring how well an inference algorithm accomplishes the specific optimization task defined by a model.

**Evaluating learning parameters.** Online LDA introduces several learning parameters: $\kappa \in (0.5, 1]$, which controls how quickly old information is forgotten; $\tau_0 \geq 0$, which downweights early iterations; and the mini-batch size $S$, which controls how many documents are used each iteration. Although online LDA converges to a stationary point for any valid $\kappa$, $\tau_0$, and $S$, the quality of this stationary point and the speed of convergence may depend on how the learning parameters are set.

We evaluated a range of settings of the learning parameters $\kappa$, $\tau_0$, and $S$ on two corpora: 352,549 documents from the journal *Nature* [3] and 100,000 documents downloaded from the English ver-

sion of Wikipedia [4]. For each corpus, we set aside a 1,000-document test set and a separate 1,000-document validation set. We then ran online LDA for five hours on the remaining documents from each corpus for $\kappa \in \{0.5, 0.6, 0.7, 0.8, 0.9, 1.0\}$, $\tau_0 \in \{1, 4, 16, 64, 256, 1024\}$, and $S \in \{1, 4, 16, 64, 256, 1024, 4096, 16384\}$, for a total of 288 runs per corpus. After five hours of CPU time, we computed perplexity on the test sets for the topics $\boldsymbol{\lambda}$ obtained at the end of each fit.

Table 1 summarizes the best settings for each corpus of $\kappa$ and $\tau_0$ for a range of settings of $S$. The supplement includes a more exhaustive summary. The best learning parameter settings for both corpora were $\kappa = 0.5$, $\tau_0 = 64$, and $S = 4096$. The best settings of $\kappa$ and $\tau_0$ are consistent across the two corpora. For mini-batch sizes from 256 to 16384 there is little difference in perplexity scores.

Several trends emerge from these results. Higher values of the learning rate $\kappa$ and the downweighting parameter $\tau_0$ lead to better performance for small mini-batch sizes $S$, but worse performance for larger values of $S$. Mini-batch sizes of at least 256 documents outperform smaller mini-batch sizes.

**Comparing batch and online on fixed corpora.** To compare batch LDA to online LDA, we evaluated held-out perplexity as a function of time on the *Nature* and Wikipedia corpora above. We tried various mini-batch sizes from 1 to 16,384, using the best learning parameters for each mini-batch size found in the previous study of the *Nature* corpus. We also evaluated batch LDA fit to a 10,000-document subset of the training corpus. We computed perplexity on a separate validation set from the test set used in the previous experiment. Each algorithm ran for 24 hours of CPU time.

Figure 2 summarizes the results. On the larger *Nature* corpus, online LDA finds a solution as good as the batch algorithm's with much less computation. On the smaller Wikipedia corpus, the online algorithm finds a better solution than the batch algorithm does. The batch algorithm converges quickly on the 10,000-document corpora, but makes less accurate predictions on held-out documents.

**True online.** To demonstrate the ability of online VB to perform in a true online setting, we wrote a Python script to continually download and analyze mini-batches of articles chosen at random from a list of approximately 3.3 million Wikipedia articles. This script can download and analyze about 60,000 articles an hour. It completed a pass through all 3.3 million articles in under three days. The amount of time needed to download an article and convert it to a vector of word counts is comparable to the amount of time that the online LDA algorithm takes to analyze it.

We ran online LDA with $\kappa = 0.5$, $\tau_0 = 1024$, and $S = 1024$. Figure 1 shows the evolution of the perplexity obtained on the held-out validation set of 1,000 Wikipedia articles by the online algorithm as a function of number of articles seen. Shown for comparison is the perplexity obtained by the online algorithm (with the same parameters) fit to only 98,000 Wikipedia articles, and that obtained by the batch algorithm fit to the same 98,000 articles.

The online algorithm outperforms the batch algorithm regardless of which training dataset is used, but it does best with access to a constant stream of novel documents. The batch algorithm's failure to outperform the online algorithm on limited data may be due to stochastic gradient's robustness to local optima [19]. The online algorithm converged after analyzing about half of the 3.3 million articles. Even one iteration of the batch algorithm over that many articles would have taken days.

## 5   Discussion

We have developed online variational Bayes (VB) for LDA. This algorithm requires only a few more lines of code than the traditional batch VB of [7], and is handily applied to massive and streaming document collections. Online VB for LDA approximates the posterior as well as previous approaches in a fraction of the time. The approach we used to derive an online version of batch VB for LDA is general (and simple) enough to apply to a wide variety of hierarchical Bayesian models.

**Acknowledgments**

D.M. Blei is supported by ONR 175-6343, NSF CAREER 0745520, AFOSR 09NL202, the Alfred P. Sloan foundation, and a grant from Google. F. Bach is supported by ANR (MGA project).

## Footnotes

[1] Although we use a deterministic procedure to compute $\gamma$ and $\phi$ given $n$ and $\boldsymbol{\lambda}$, this analysis can also be applied if $\gamma$ and $\phi$ are optimized using a randomized algorithm. We address this case in the supplement.

[2]Open-source Python implementations of batch and online LDA can be found at `http://www.cs.princeton.edu/~mdhoffma`.

[3]For the Nature articles, we removed all words not in a pruned vocabulary of 4,253 words.

[4]For the Wikipedia articles, we removed all words not from a fixed vocabulary of 7,995 common words. This vocabulary was obtained by removing words less than 3 characters long from a list of the 10,000 most common words in Project Gutenberg texts obtained from http://en.wiktionary.org/wiki/Wiktionary:Frequency_lists.

# References

[1] M. Braun and J. McAuliffe. Variational inference for large-scale models of discrete choice. *arXiv*, (0712.2526), 2008.

[2] D. Blei and M. Jordan. Variational methods for the Dirichlet process. In *Proc. 21st Int'l Conf. on Machine Learning*, 2004.

[3] A. Asuncion, M. Welling, P. Smyth, and Y.W. Teh. On smoothing and inference for topic models. In *Proceedings of the 25th Conference on Uncertainty in Artificial Intelligence*, 2009.

[4] D. Newman, A. Asuncion, P. Smyth, and M. Welling. Distributed inference for latent Dirichlet allocation. In *Neural Information Processing Systems*, 2007.

[5] Feng Yan, Ningyi Xu, and Yuan Qi. Parallel inference for latent Dirichlet allocation on graphics processing units. In *Advances in Neural Information Processing Systems 22*, pages 2134–2142, 2009.

[6] L. Bottou and O. Bousquet. The tradeoffs of large scale learning. In *Advances in Neural Information Processing Systems*, volume 20, pages 161–168. NIPS Foundation (http://books.nips.cc), 2008.

[7] D. Blei, A. Ng, and M. Jordan. Latent Dirichlet allocation. *Journal of Machine Learning Research*, 3:993–1022, January 2003.

[8] Hanna Wallach, David Mimno, and Andrew McCallum. Rethinking lda: Why priors matter. In *Advances in Neural Information Processing Systems 22*, pages 1973–1981, 2009.

[9] W. Buntine. Variational extentions to EM and multinomial PCA. In *European Conf. on Machine Learning*, 2002.

[10] J. Mairal, F. Bach, J. Ponce, and G. Sapiro. Online learning for matrix factorization and sparse coding. *Journal of Machine Learning Research*, 11(1):19–60, 2010.

[11] L. Yao, D. Mimno, and A. McCallum. Efficient methods for topic model inference on streaming document collections. In *KDD 2009: Proc. 15th ACM SIGKDD int'l Conf. on Knowledge discovery and data mining*, pages 937–946, 2009.

[12] M. Jordan, Z. Ghahramani, T. Jaakkola, and L. Saul. Introduction to variational methods for graphical models. *Machine Learning*, 37:183–233, 1999.

[13] H. Attias. A variational Bayesian framework for graphical models. In *Advances in Neural Information Processing Systems 12*, 2000.

[14] A. Dempster, N. Laird, and D. Rubin. Maximum likelihood from incomplete data via the EM algorithm. *Journal of the Royal Statistical Society, Series B*, 39:1–38, 1977.

[15] L. Bottou and N. Murata. Stochastic approximations and efficient learning. *The Handbook of Brain Theory and Neural Networks, Second edition. The MIT Press, Cambridge, MA*, 2002.

[16] M.A. Sato. Online model selection based on the variational Bayes. *Neural Computation*, 13(7):1649–1681, 2001.

[17] P. Liang and D. Klein. Online EM for unsupervised models. In *Proc. Human Language Technologies: The 2009 Annual Conference of the North American Chapter of the Association for Computational Linguistics*, pages 611–619, 2009.

[18] H. Robbins and S. Monro. A stochastic approximation method. *The Annals of Mathematical Statistics*, 22(3):400–407, 1951.

[19] L. Bottou. *Online learning and stochastic approximations*. Cambridge University Press, Cambridge, UK, 1998.

[20] R.M. Neal and G.E. Hinton. A view of the EM algorithm that justifies incremental, sparse, and other variants. *Learning in graphical models*, 89:355–368, 1998.

[21] M.A. Sato and S. Ishii. On-line EM algorithm for the normalized Gaussian network. *Neural Computation*, 12(2):407–432, 2000.

[22] T. Griffiths and M. Steyvers. Finding scientific topics. *Proc. National Academy of Science*, 2004.

[23] X. Song, C.Y. Lin, B.L. Tseng, and M.T. Sun. Modeling and predicting personal information dissemination behavior. In *KDD 2005: Proc. 11th ACM SIGKDD int'l Conf. on Knowledge discovery and data mining*. ACM, 2005.

[24] K.R. Canini, L. Shi, and T.L. Griffiths. Online inference of topics with latent Dirichlet allocation. In *Proceedings of the International Conference on Artificial Intelligence and Statistics*, volume 5, 2009.

[25] J. Chang, J. Boyd-Graber, S. Gerrish, C. Wang, and D. Blei. Reading tea leaves: How humans interpret topic models. In *Advances in Neural Information Processing Systems 21 (NIPS)*, 2009.

